# A Convergence Proof for the Softassign Quadratic Assignment Algorithm

**Anand Rangarajan**
Department of Diagnostic Radiology
Yale University School of Medicine
New Haven, CT 06520-8042
e-mail: anand@noodle.med.yale.edu

**Alan Yuille**
Smith-Kettlewell Eye Institute
2232 Webster Street
San Francisco, CA 94115
e-mail: yuille@skivs.ski.org

**Steven Gold**
CuraGen Corporation
322 East Main Street
Branford, CT 06405
e-mail: gold-steven@cs.yale.edu

**Eric Mjolsness**
Dept. of Comp. Sci. and Engg.
Univ. of California San Diego (UCSD)
La Jolla, CA 92093-0114
e-mail: emj@cs.ucsd.edu

## Abstract

The softassign quadratic assignment algorithm has recently emerged as an effective strategy for a variety of optimization problems in pattern recognition and combinatorial optimization. While the effectiveness of the algorithm was demonstrated in thousands of simulations, there was no known proof of convergence. Here, we provide a proof of convergence for the most general form of the algorithm.

## 1 Introduction

Recently, a new neural optimization algorithm has emerged for solving quadratic assignment like problems [4, 2]. Quadratic assignment problems (QAP) are characterized by quadratic objectives with the variables obeying permutation matrix constraints. Problems that roughly fall into this class are TSP, graph partitioning (GP) and graph matching. The new algorithm is based on the softassign procedure which guarantees the satisfaction of the doubly stochastic matrix constraints (resulting from a "neural" style relaxation of the permutation matrix constraints). While the effectiveness of the softassign procedure has been demonstrated via thousands

of simulations, no proof of convergence was ever shown.

Here, we show a proof of convergence for the softassign quadratic assignment algorithm. The proof is based on algebraic transformations of the original objective and on the non-negativity of the Kullback-Leibler measure. A central requirement of the proof is that the softassign procedure always returns a doubly stochastic matrix. After providing a general criterion for convergence, we separately analyze the cases of TSP and graph matching.

## 2 Convergence proof

The deterministic annealing quadratic assignment objective function is written as [4, 5]:

$$
E_{\text{qap}}(M, \mu, \nu) = -\frac{1}{2} \sum_{aibj} C_{ai;bj} M_{ai} M_{bj} + \sum_a \mu_a (\sum_i M_{ai} - 1) + \sum_i \nu_i (\sum_a M_{ai} - 1)
$$
$$
-\frac{\gamma}{2} \sum_{ai} M_{ai}^2 + \frac{1}{\beta} \sum_{ai} M_{ai} \log M_{ai} \quad (1)
$$

Here $M$ is the desired $N \times N$ permutation matrix. This form of the energy function has a *self-amplification* term with a parameter $\gamma$, two Lagrange parameters $\mu$ and $\nu$ for constraint satisfaction, an $x \log x$ barrier function which ensures positivity of $M_{ai}$ and a deterministic annealing control parameter $\beta$. The QAP benefit matrix $C_{ai;bj}$ is preset based on the chosen problem, for example, graph matching or TSP. In the following deterministic annealing pseudocode $\beta_0$ and $\beta_f$ are the initial and final values of $\beta$, $\beta_r$ is the rate at which $\beta$ is increased, $I_B$ is an iteration cap and $\xi$ is an $N \times N$ matrix of small positive-valued random numbers.

Initialize $\beta$ to $\beta_0$, $M_{ai}$ to $\frac{1}{N} + \xi_{ai}$

**Begin A: Deterministic annealing.** Do A until $\beta \geq \beta_f$

    **Begin B: Relaxation.** Do B until all $M_{ai}$ converge or number of iterations $> I_B$

    $Q_{ai} \leftarrow \sum_{bj} C_{ai;bj} M_{bj} + \gamma M_{ai}$

        **Begin Softassign:**

    $M_{ai} \leftarrow \exp(\beta Q_{ai})$

        **Begin C: Sinkhorn.** Do C until all $M_{ai}$ converge

        Update $M_{ai}$ by normalizing the rows:

    $M_{ai} \leftarrow \frac{M_{ai}}{\sum_i M_{ai}}$

        Update $M_{ai}$ by normalizing the columns:

    $M_{ai} \leftarrow \frac{M_{ai}}{\sum_a M_{ai}}$

        **End C**

        **End Softassign**

    **End B**

    $\beta \leftarrow \beta_r \beta$

**End A**

The softassign is used for constraint satisfaction. The softassign is based on Sinkhorn's theorem [4] but can be independently derived as coordinate ascent on the Lagrange parameters $\mu$ and $\nu$. Sinkhorn's theorem ensures that we obtain a doubly stochastic matrix by the simple process of alternating row and column normalizations. The QAP algorithm above was developed using the graduated assignment heuristic [1] with no proof of convergence until now.

We simplify the objective function in (1) by collecting together all terms quadratic in $M_{ai}$. This is achieved by defining

$$C^{(\gamma)}_{ai;ai} = C_{ai;ai} + \gamma. \tag{2}$$

Then we use an *algebraic transformation* [3] to transform the quadratic form into a more manageable linear form:

$$-\frac{X^2}{2} \to \min_{\sigma} \left( -X\sigma + \frac{1}{2}\sigma^2 \right) \tag{3}$$

Application of the algebraic transformation (in a vectorized form) to the quadratic term in (1) yields:

$$E_{\text{qap}}(M,\sigma,\mu,\nu) = -\sum_{aibj} C^{(\gamma)}_{ai;bj} M_{ai}\sigma_{bj} + \frac{1}{2}\sum_{aibj} C^{(\gamma)}_{ai;bj}\sigma_{ai}\sigma_{bj}$$

$$+ \sum_{a}\mu_a(\sum_i M_{ai} - 1) + \sum_i \nu_i(\sum_a M_{ai} - 1) + \frac{1}{\beta}\sum_{ai} M_{ai}\log M_{ai} \tag{4}$$

Extremizing (4) w.r.t. $\sigma$, we get

$$\sum_{bj} C^{(\gamma)}_{ai;bj} M_{bj} = \sum_{bj} C^{(\gamma)}_{ai;bj}\sigma_{bj} \Rightarrow \sigma_{ai} = M_{ai} \tag{5}$$

is a minimum, provided certain conditions hold which we specify below.

In the first part of the proof, we show that setting $\sigma_{ai} = M_{ai}$ is guaranteed to decrease the energy function. Restated, we require that

$$\sigma_{ai} = M_{ai} = \arg\min_{\sigma}\left( -\sum_{aibj} C^{(\gamma)}_{ai;bj} M_{ai}\sigma_{bj} + \frac{1}{2}\sum_{aibj} C^{(\gamma)}_{ai;bj}\sigma_{ai}\sigma_{bj} \right) \tag{6}$$

If $C^{(\gamma)}_{ai;bj}$ is positive definite in the subspace spanned by $M$, then $\sigma_{ai} = M_{ai}$ is a minimum of the energy function $-\sum_{aibj} C^{(\gamma)}_{ai;bj} M_{ai}\sigma_{bj} + \frac{1}{2}\sum_{aibj} C^{(\gamma)}_{ai;bj}\sigma_{ai}\sigma_{bj}$.

At this juncture, we make a crucial assumption that considerably simplifies the proof. Since this assumption is central, we formally state it here: "*M is always constrained to be a doubly stochastic matrix.*" In other words, for our proof of convergence, we require the softassign algorithm to return a doubly stochastic matrix (as Sinkhorn's theorem guarantees that it will) instead of a matrix which is merely close to being doubly stochastic (based on some reasonable metric). We also require the variable $\sigma$ to be a doubly stochastic matrix.

Since $M$ is always constrained to be a doubly stochastic matrix, $C^{(\gamma)}_{ai;bj}$ is required to be positive definite in the linear subspace of rows and columns of $M$ summing to

one. The value of $\gamma$ should be set high enough such that $C^{(\gamma)}_{ai;bj}$ does not have any negative eigenvalues in the subspace spanned by the row and column constraints. This is the same requirement imposed in [5] to ensure that we obtain a permutation matrix at zero temperature.

To derive a more explicit criterion for $\gamma$, we first define a matrix $r$ in the following manner:

$$r \overset{\text{def}}{=} I_N - \frac{1}{N} ee' \tag{7}$$

where $I_N$ is the $N \times N$ identity matrix, $e$ is the vector of all ones and the "prime" indicates a transpose operation. The matrix $r$ has the property that any vector $rs$ with $s$ arbitrary will sum to zero. We would like to extend such a property to cover matrices whose row and column sums stay fixed. To achieve this, take the Kronecker product of $r$ with itself:

$$R \overset{\text{def}}{=} r \otimes r \tag{8}$$

$R$ has the property that it will annihilate all row and column sums. Form a vector $m$ by concatenating all the columns of the matrix $M$ together into a single column $[m = \text{vec}(M)]$. Then the vector $Rm$ has the equivalent property of the "rows" and "columns" summing to zero. Hence the matrix $RC^{(\gamma)}R$ (where $C^{(\gamma)}$ is the matrix equivalent of $C^{(\gamma)}_{ai;bj}$) satisfies the criterion of annihilated row and column sums in any quadratic form; $m'RC^{(\gamma)}Rm = (Rm)'C^{(\gamma)}(Rm)$.

The parameter $\gamma$ is chosen such that all eigenvalues of $RC^{(\gamma)}R$ are positive:

$$\gamma = -\min_{\lambda} \lambda(RCR) + \epsilon \tag{9}$$

where $\epsilon > 0$ is a small quantity. Note that $C$ is the original QAP benefit matrix whereas $C^{(\gamma)}$ is the augmented matrix of (2). We cannot always efficiently compute the largest negative eigenvalue of the matrix $RCR$. Since the original $C_{ai;bj}$ is four dimensional, the dimensions of $RCR$ are $N^2 \times N^2$ where $N$ is the number of elements in one set. Fortunately, as we show later, for specific problems it's possible to break up $RCR$ into its constituents thereby making the calculation of the largest negative eigenvalue of $RCR$ more efficient. We return to this point in Section 3.

The second part of the proof involves demonstrating that the softassign operation also decreases the objective in (4). (Note that the two Lagrange parameters $\mu$ and $\nu$ are specified by the softassign algorithm [4]).

$$M = \text{Softassign}(Q, \beta) \text{ where } Q_{ai} = \sum_{bj} C^{(\gamma)}_{ai;bj} \sigma_{bj} \tag{10}$$

Recall that the step immediately preceding the softassign operation sets $\sigma_{ai} = M_{ai}$. We are therefore justified in referring to $\sigma_{ai}$ as the "old" value of $M_{ai}$. For convergence, we have to show that $E_{\text{qap}}(\sigma, \sigma) \geq E_{\text{qap}}(M, \sigma)$ in (4).

Minimizing (4) w.r.t. $M_{ai}$, we get

$$\frac{1}{\beta} \log M_{ai} = \sum_{bj} C^{(\gamma)}_{ai;bj} \sigma_{bj} - (\mu_a + \nu_i) - \frac{1}{\beta} \tag{11}$$

From (11), we see that

$$\frac{1}{\beta} \sum_{ai} M_{ai} \log M_{ai} = \sum_{aibj} C_{ai;bj}^{(\gamma)} M_{ai} \sigma_{bj} - \sum_a \mu_a \sum_i M_{ai} - \sum_i \nu_i \sum_a M_{ai} - \frac{1}{\beta} \sum_{ai} M_{ai}$$

$$(12)$$

and

$$\frac{1}{\beta} \sum_{ai} \sigma_{ai} \log M_{ai} = \sum_{aibj} C_{ai;bj}^{(\gamma)} \sigma_{ai} \sigma_{bj} - \sum_a \mu_a \sum_i \sigma_{ai} - \sum_i \nu_i \sum_a \sigma_{ai} - \frac{1}{\beta} \sum_{ai} \sigma_{ai} \quad (13)$$

From (12) and (13), we get (after some algebraic manipulations)

$$E_{\mathrm{qap}}(\sigma, \sigma) - E_{\mathrm{qap}}(M, \sigma) =$$

$$-\sum_{aibj} C_{ai;bj}^{(\gamma)} \sigma_{ai} \sigma_{bj} - \left( -\sum_{aibj} C_{ai;bj}^{(\gamma)} M_{ai} \sigma_{bj} \right) + \frac{1}{\beta} \sum_{ai} \sigma_{ai} \log \sigma_{ai} - \frac{1}{\beta} \sum_{ai} M_{ai} \log M_{ai}$$

$$= \frac{1}{\beta} \sum_{ai} \sigma_{ai} \log \frac{\sigma_{ai}}{M_{ai}} \geq 0$$

by the non-negativity of the Kullback-Leibler measure. We have shown that the change in energy *after $\sigma$ has been initialized with the "old" value of $M$* is non-negative. We require that $\sigma$ and $M$ are always doubly stochastic via the action of the softassign operation. Consequently, the terms involving the Lagrange parameters $\mu$ and $\nu$ can be eliminated from the energy function (4). Setting $\sigma = M$ followed by the softassign operation decreases the objective in (4) after excluding the terms involving the Lagrange parameters.

We summarize the essence of the proof to bring out the salient points. At each temperature, the quadratic assignment algorithm executes the following steps until convergence is established.

**Step 1:** $\sigma_{ai} \leftarrow M_{ai}$.

**Step 2:**

   **Step 2a:** $Q_{ai} \leftarrow \sum_{bj} C_{ai;bj}^{(\gamma)} \sigma_{bj}$.
   **Step 2b:** $M \leftarrow \mathrm{Softassign}(Q, \beta)$.

Return to Step 1 until convergence.

Our proof is based on demonstrating that an appropriately designed energy function decreases in both Step 1 and Step 2 (at fixed temperature). This energy function is Equation (4) after excluding the Lagrange parameter terms.

**Step 1:** Energy decreases due to the positive definiteness of $C_{ai;bj}^{(\gamma)}$ in the *linear* subspace spanned by the row and column constraints. $\gamma$ has to be set high enough for this statement to be true.

**Step 2:** Energy decreases due to the non-negativity of the Kullback-Leibler measure and due to the restriction that $M$ (and $\sigma$) are doubly stochastic.

## 3  Applications

### 3.1  Quadratic Assignment

The QAP benefit matrix is chosen such that the softassign algorithm will not converge without adding the $\gamma$ term in (1). To achieve this, we randomly picked a unit vector $v$ of dimension $N^2$. The benefit matrix $C$ is set to $-vv'$. Since $C$ has only one negative eigenvalue, the softassign algorithm cannot possibly converge. We ran the softassign algorithm with $\beta_0 = 1$, $\beta_r = 0.9$ and $\gamma = 0$. The energy difference plot on the left in Figure 1 shows the energy never decreasing with increasing iteration number. Next, we followed the recipe for setting $\gamma$ exactly as in Section 2. After projecting $C$ into the subspace of the row and column constraints, we calculated the largest negative eigenvalue of the matrix $RCR$ which turned out to be -0.8152. We set $\gamma$ to 0.8162 ($\epsilon = 0.001$) and reran the softassign algorithm. The energy difference plot shows (Figure 1) that the energy never increases. We have shown that a proper choice of $\gamma$ leads to a convergent algorithm.

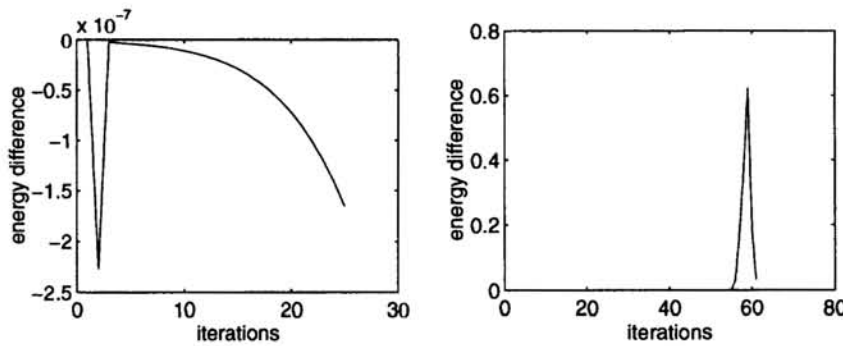

Figure 1: **Energy difference plot.** Left: $\gamma = 0$ and Right: $\gamma = 0.8162$. While the change in energy is always negative when $\gamma = 0$, it is always non-negative when $\gamma = 0.8162$. The negative energy difference (on the left) implies that the energy function increases whereas the non-negative energy difference (on the right) implies that the energy function never increases.

### 3.2  TSP

The TSP objective function is written as follows: Given $N$ cities,

$$E_{\text{tsp}}(M) = \sum_{aij} d_{ij} M_{ai} M_{(a \oplus 1)j} = \text{trace}(DM'TM) \tag{14}$$

where the symbol $\oplus$ is used to indicate that the summation in (14) is taken modulo $N$, $d_{ij}$ ($D$) is the inter-city distance matrix and $M$ is the desired permutation matrix. $T$ is a matrix whose $(i, j)^{\text{th}}$ entry is $\delta_{(i \oplus 1)j}$ ($\delta_{ij}$ is the Kronecker delta function). Equation (14) is transformed into the $m'Cm$ form:

$$E_{\text{tsp}}(m) = \text{trace}(m'(D \otimes T)m) \tag{15}$$

where $m = \text{vec}(M)$. We identify our general matrix $C$ with $-2D \otimes T$.

For convergence, we require the largest eigenvalue of

$$-RCR = 2(r \otimes r)(D \otimes T)(r \otimes r) = 2(rDr) \otimes (rTr) = 2(rDr) \otimes (rT) \qquad (16)$$

The eigenvalues of $rT$ are bounded by unity. The eigenvalues of $rDr$ will depend on the form of $D$. Even in Euclidean TSP the values will depend on whether the Euclidean distance or the distance squared between the cities is used.

### 3.3  Graph Matching

The graph matching objective function is written as follows: Given $N_1$ and $N_2$ node graphs with adjacency matrices $G$ and $g$ respectively,

$$E_{\mathrm{gm}}(M) = -\frac{1}{2}\sum_{aibj} C_{ai;bj} M_{ai} M_{bj} \qquad (17)$$

where $C_{ai;bj} = 1 - 3|G_{ab} - g_{ij}|$ is the compatibility matrix [1]. The matching constraints are somewhat different from TSP due to the presence of slack variables [1]. This makes no difference however to our projection operators. We add an extra row and column of zeros to $g$ and $G$ in order to handle the slack variable case. Now $G$ is $(N_1 + 1) \times (N_1 + 1)$ and $g$ is $(N_2 + 1) \times (N_2 + 1)$. Equation (17) can be readily transformed into the $m'Cm$ form. Our projection apparatus remains unchanged. For convergence, we require the largest negative eigenvalue of $RCR$.

## 4  Conclusion

We have derived a convergence proof for the softassign quadratic assignment algorithm and specialized to the cases of TSP and graph matching. An extension to graph partitioning follows along the same lines as graph matching. Central to our proof is the requirement that the QAP matrix $M$ is always doubly stochastic. As a by-product, the convergence proof yields a criterion by which the free self-amplification parameter $\gamma$ is set. We believe that the combination of good theoretical properties and experimental success of the softassign algorithm make it the technique of choice for quadratic assignment neural optimization.

## References

[1] S. Gold and A. Rangarajan. A graduated assignment algorithm for graph matching. *IEEE Transactions on Pattern Analysis and Machine Intelligence*, 18(4):377–388, 1996.

[2] S. Gold and A. Rangarajan. Softassign versus softmax: Benchmarks in combinatorial optimization. In *Advances in Neural Information Processing Systems 8*, pages 626–632. MIT Press, 1996.

[3] E. Mjolsness and C. Garrett. Algebraic transformations of objective functions. *Neural Networks*, 3:651–669, 1990.

[4] A. Rangarajan, S. Gold, and E. Mjolsness. A novel optimizing network architecture with applications. *Neural Computation*, 8(5):1041–1060, 1996.

[5] A. L. Yuille and J. J. Kosowsky. Statistical physics algorithms that converge. *Neural Computation*, 6(3):341–356, May 1994.